# Efficient Out-of-Sample Extension of Dominant-Set Clusters

**Massimiliano Pavan and Marcello Pelillo**
Dipartimento di Informatica, Università Ca' Foscari di Venezia
Via Torino 155, 30172 Venezia Mestre, Italy
{pavan,pelillo}@dsi.unive.it

## Abstract

Dominant sets are a new graph-theoretic concept that has proven to be relevant in pairwise data clustering problems, such as image segmentation. They generalize the notion of a maximal clique to edge-weighted graphs and have intriguing, non-trivial connections to continuous quadratic optimization and spectral-based grouping. We address the problem of grouping out-of-sample examples after the clustering process has taken place. This may serve either to drastically reduce the computational burden associated to the processing of very large data sets, or to efficiently deal with dynamic situations whereby data sets need to be updated continually. We show that the very notion of a dominant set offers a simple and efficient way of doing this. Numerical experiments on various grouping problems show the effectiveness of the approach.

## 1   Introduction

Proximity-based, or pairwise, data clustering techniques are gaining increasing popularity over traditional central grouping techniques, which are centered around the notion of "feature" (see, e.g., [3, 12, 13, 11]). In many application domains, in fact, the objects to be clustered are not naturally representable in terms of a vector of features. On the other hand, quite often it is possible to obtain a measure of the similarity/dissimilarity between objects. Hence, it is natural to map (possibly implicitly) the data to be clustered to the nodes of a weighted graph, with edge weights representing similarity or dissimilarity relations. Although such a representation lacks geometric notions such as scatter and centroid, it is attractive as no feature selection is required and it keeps the algorithm generic and independent from the actual data representation. Further, it allows one to use non-metric similarities and it is applicable to problems that do not have a natural embedding to a uniform feature space, such as the grouping of structural or graph-based representations.

We have recently developed a new framework for pairwise data clustering based on a novel graph-theoretic concept, that of a *dominant set*, which generalizes the notion of a maximal clique to edge-weighted graphs [7, 9]. An intriguing connection between dominant sets and the solutions of a (continuous) quadratic optimization problem makes them related in a non-trivial way to spectral-based cluster notions, and allows one to use straightforward dynamics from evolutionary game theory to determine them [14]. A nice feature of this framework is that it naturally provides a principled measure of a cluster's cohesiveness as well as a measure of a vertex participation to its assigned group. It also allows one to obtain "soft" partitions of the input data, by allowing a point to belong to more than one cluster. The approach has proven to be a powerful one when applied to problems such as intensity, color, and texture segmentation, or visual database organization, and is competitive with

spectral approaches such as normalized cut [7, 8, 9].

However, a typical problem associated to pairwise grouping algorithms in general, and hence to the dominant set framework in particular, is the scaling behavior with the number of data. On a dataset containing $N$ examples, the number of potential comparisons scales with $O(N^2)$, thereby hindering their applicability to problems involving very large data sets, such as high-resolution imagery and spatio-temporal data. Moreover, in applications such as document classification or visual database organization, one is confronted with a dynamic environment which continually supplies the algorithm with newly produced data that have to be grouped. In such situations, the trivial approach of recomputing the complete cluster structure upon the arrival of any new item is clearly unfeasible.

Motivated by the previous arguments, in this paper we address the problem of efficiently assigning out-of-sample, unseen data to one or more previously determined clusters. This may serve either to substantially reduce the computational burden associated to the processing of very large (though static) data sets, by extrapolating the complete grouping solution from a small number of samples, or to deal with dynamic situations whereby data sets need to be updated continually. There is no straightforward way of accomplishing this within the pairwise grouping paradigm, short of recomputing the complete cluster structure. Recent sophisticated attempts to deal with this problem use optimal embeddings [11] and the Nyström method [1, 2]. By contrast, we shall see that the very notion of a dominant set, thanks to its clear combinatorial properties, offers a simple and efficient solution to this problem. The basic idea consists of computing, for any new example, a quantity which measures the degree of cluster membership, and we provide simple approximations which allow us to do this in linear time and space, with respect to the cluster size. Our classification schema inherits the main features of the dominant set formulation, i.e., the ability of yielding a soft classification of the input data and of providing principled measures for cluster membership and cohesiveness.

Numerical experiments show that the strategy of first grouping a small number of data items and then classifying the out-of-sample instances using our prediction rule is clearly successful as we are able to obtain essentially the same results as the dense problem in much less time. We also present results on high-resolution image segmentation problems, a task where the dominant set framework would otherwise be computationally impractical.

## 2 Dominant Sets and Their Continuous Characterization

We represent the data to be clustered as an undirected edge-weighted (*similarity*) graph with no self-loops $G = (V, E, w)$, where $V = \{1, \ldots, n\}$ is the vertex set, $E \subseteq V \times V$ is the edge set, and $w : E \rightarrow \mathbb{R}_+^*$ is the (positive) weight function. Vertices in $G$ correspond to data points, edges represent neighborhood relationships, and edge-weights reflect similarity between pairs of linked vertices. As customary, we represent the graph $G$ with the corresponding weighted adjacency (or similarity) matrix, which is the $n \times n$ nonnegative, symmetric matrix $A = (a_{ij})$ defined as:

$$a_{ij} = \begin{cases} w(i,j), & \text{if } (i,j) \in E \\ 0, & \text{otherwise}. \end{cases}$$

Let $S \subseteq V$ be a non-empty subset of vertices and $i \in V$. The *(average) weighted degree* of $i$ w.r.t. $S$ is defined as:

$$\text{awdeg}_S(i) = \frac{1}{|S|} \sum_{j \in S} a_{ij} \tag{1}$$

where $|S|$ denotes the cardinality of $S$. Moreover, if $j \notin S$ we define $\phi_S(i,j) = a_{ij} - \text{awdeg}_S(i)$ which is a measure of the similarity between nodes $j$ and $i$, with respect to the average similarity between node $i$ and its neighbors in $S$.

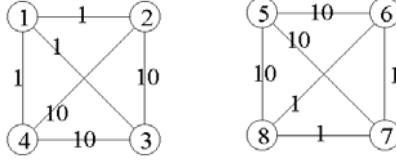

Figure 1: An example edge-weighted graph. Note that $w_{\{1,2,3,4\}}(1) < 0$ and this reflects the fact that vertex 1 is loosely coupled to vertices 2, 3 and 4. Conversely, $w_{\{5,6,7,8\}}(5) > 0$ and this reflects the fact that vertex 5 is tightly coupled with vertices 6, 7, and 8.

Let $S \subseteq V$ be a non-empty subset of vertices and $i \in S$. The *weight* of $i$ w.r.t. $S$ is

$$
w_S(i) = \begin{cases} 1, & \text{if } |S| = 1 \\ \sum_{j \in S \setminus \{i\}} \phi_{S \setminus \{i\}}(j,i)\, w_{S \setminus \{i\}}(j), & \text{otherwise} \end{cases}
\tag{2}
$$

while the *total weight* of $S$ is defined as:

$$
W(S) = \sum_{i \in S} w_S(i).
\tag{3}
$$

Intuitively, $w_S(i)$ gives us a measure of the overall similarity between vertex $i$ and the vertices of $S \setminus \{i\}$ with respect to the overall similarity among the vertices in $S \setminus \{i\}$, with positive values indicating high internal coherency (see Fig. 1).

A non-empty subset of vertices $S \subseteq V$ such that $W(T) > 0$ for any non-empty $T \subseteq S$, is said to be *dominant* if:

1. $w_S(i) > 0$, for all $i \in S$
2. $w_{S \cup \{i\}}(i) < 0$, for all $i \notin S$.

The two previous conditions correspond to the two main properties of a cluster: the first regards internal homogeneity, whereas the second regards external inhomogeneity. The above definition represents our formalization of the concept of a cluster in an edge-weighted graph.

Now, consider the following quadratic program, which is a generalization of the so-called Motzkin-Straus program [5] (here and in the sequel a dot denotes the standard scalar product between vectors):

$$
\begin{array}{ll}
\text{maximize} & f(\mathbf{x}) = \mathbf{x} \cdot A\mathbf{x} \\
\text{subject to} & \mathbf{x} \in \Delta_n
\end{array}
\tag{4}
$$

where

$$
\Delta_n = \{\mathbf{x} \in \mathbb{R}^n \ : \ x_i \geq 0 \text{ for all } i \in V \text{ and } \mathbf{e} \cdot \mathbf{x} = 1\}
$$

is the standard simplex of $\mathbb{R}^n$, and $\mathbf{e}$ is a vector of appropriate length consisting of unit entries (hence $\mathbf{e} \cdot \mathbf{x} = \sum_i x_i$). The *support* of a vector $\mathbf{x} \in \Delta_n$ is defined as the set of indices corresponding to its positive components, that is $\sigma(\mathbf{x}) = \{i \in V \ : \ x_i > 0\}$. The following theorem, proved in [7], establishes an intriguing connection between dominant sets and local solutions of program (4).

**Theorem 1** *If $S$ is a dominant subset of vertices, then its (weighted) characteristics vector $\mathbf{x}^S$, which is the vector of $\Delta_n$ defined as*

$$
x_i^S = \begin{cases} \frac{w_S(i)}{W(S)}, & \text{if } i \in S \\ 0, & \text{otherwise} \end{cases}
\tag{5}
$$

*is a strict local solution of program (4). Conversely, if $\mathbf{x}$ is a strict local solution of program (4) then its support $S = \sigma(\mathbf{x})$ is a dominant set, provided that $w_{S \cup \{i\}}(i) \neq 0$ for all $i \notin S$.*

The condition that $w_{S \cup \{i\}}(i) \neq 0$ for all $i \notin S = \sigma(\mathbf{x})$ is a technicality due to the presence of "spurious" solutions in (4) which is, at any rate, a non-generic situation.

By virtue of this result, we can find a dominant set by localizing a local solution of program (4) with an appropriate continuous optimization technique, such as replicator dynamics from evolutionary game theory [14], and then picking up its support. Note that the components of the weighted characteristic vectors give us a natural measure of the participation of the corresponding vertices in the cluster, whereas the value of the objective function measures the cohesiveness of the class. In order to get a partition of the input data into coherent groups, a simple approach is to iteratively finding a dominant set and then removing it from the graph, until all vertices have been grouped (see [9] for a hierarchical extension of this framework). On the other hand, by finding all dominant sets, i.e., local solutions of (4), of the original graph, one can obtain a "soft" partition of the dataset, whereby clusters are allowed to overlap. Finally, note that spectral clustering approaches such as, e.g., [10, 12, 13] lead to similar, though intrinsically different, optimization problems.

## 3  Predicting Cluster Membership for Out-of-Sample Data

Suppose we are given a set $V$ of $n$ unlabeled items and let $G = (V, E, w)$ denote the corresponding similarity graph. After determining the dominant sets (i.e., the clusters) for these original data, we are next supplied with a set $V'$ of $k$ new data items, together with all $kn$ pairwise affinities between the old and the new data, and are asked to assign each of them to one or possibly more previously determined clusters. We shall denote by $\hat{G} = (\hat{V}, \hat{E}, \hat{w})$, with $\hat{V} = V \cup V'$, the similarity graph built upon all the $n + k$ data. Note that in our approach we do not need the $\binom{k}{2}$ affinities between the new points, which is a nice feature as in most applications $k$ is typically very large. Technically, $\hat{G}$ is a *supergraph* of $G$, namely a graph having $V \subseteq \hat{V}$, $E \subseteq \hat{E}$ and $w(i, j) = \hat{w}(i, j)$ for all $(i, j) \in E$.

Let $S \subseteq V$ be a subset of vertices which is dominant in the original graph $G$ and let $i \in \hat{V} \setminus V$ a new data point. As pointed out in the previous section, the sign of $w_{S \cup \{i\}}(i)$ provides an indication as to whether $i$ is tightly or loosely coupled with the vertices in $S$ (the condition $w_{S \cup \{i\}}(i) = 0$ corresponds to a non-generic boundary situation that does not arise in practice and will therefore be ignored).[1] Accordingly, it is natural to propose the following rule for predicting cluster membership of unseen data:

$$\text{if } w_{S \cup \{i\}}(i) > 0, \text{ then assign vertex } i \text{ to cluster } S. \qquad (6)$$

Note that, according to this rule, the same point can be assigned to more than one class, thereby yielding a soft partition of the input data. To get a hard partition one can use the cluster membership approximation measures we shall discuss below. Note that it may also happen for some instance $i$ that no cluster $S$ satisfies rule (6), in which case the point gets unclassified (or assigned to an "outlier" group). This should be interpreted as an indication that either the point is too noisy or that the cluster formation process was inaccurate. In our experience, however, this situation arises rarely.

A potential problem with the previous rule is its computational complexity. In fact, a direct application of formula (2) to compute $w_{S \cup \{i\}}(i)$ is clearly infeasible due to its recursive nature. On the other hand, using a characterization given in [7, Lemma 1] would also be expensive since it would involve the computation of a determinant. The next result allows us to compute the sign of $w_{S \cup \{i\}}(i)$ in linear time and space, with respect to the size of $S$.

**Proposition 1** *Let $G = (V, E, w)$ be an edge-weighted (similarity) graph, $A = (a_{ij})$ its weighted adjacency matrix, and $S \subseteq V$ a dominant set of $G$ with characteristic vector*

$\mathbf{x}^S$. Let $\hat{G} = (\hat{V}, \hat{E}, \hat{w})$ be a supergraph of $G$ with weighted adjacency matrix $\hat{A} = (\hat{a}_{ij})$. Then, for all $i \in \hat{V} \setminus V$, we have:

$$\mathrm{w}_{S \cup \{i\}}(i) > 0 \quad \Leftrightarrow \quad \sum_{h \in S} \hat{a}_{hi} x_h^S > f(\mathbf{x}^S) \tag{7}$$

*Proof.* From Theorem 1, $\mathbf{x}^S$ is a strict local solution of program (4) and hence it satisfies the Karush-Kuhn-Tucker (KKT) equality conditions, i.e., the first-order necessary equality conditions for local optimality [4]. Now, let $\hat{n} = |\hat{V}|$ be the cardinality of $\hat{V}$ and let $\hat{\mathbf{x}}^S$ be the ($\hat{n}$-dimensional) characteristic vector of $S$ in $\hat{G}$, which is obtained by padding $\mathbf{x}^S$ with zeros. It is immediate to see that $\hat{\mathbf{x}}^S$ satisfies the KKT equality conditions for the problem of maximizing $\hat{f}(\hat{\mathbf{x}}) = \hat{\mathbf{x}} \cdot \hat{A}\hat{\mathbf{x}}$, subject to $\hat{\mathbf{x}} \in \Delta_{\hat{n}}$. Hence, from Lemma 2 of [7] we have for all $i \in \hat{V} \setminus V$:

$$\frac{\mathrm{w}_{S \cup \{i\}}(i)}{\mathrm{W}(S)} = \sum_{h \in S} (\hat{a}_{hi} - a_{hj}) x_h^S \tag{8}$$

for any $j \in S$. Now, recall that the KKT equality conditions for program (4) imply $\sum_{h \in S} a_{hj} x_h^S = \mathbf{x}^S \cdot A\mathbf{x}^S = f(\mathbf{x}^S)$ for any $j \in S$ [7]. Hence, the proposition follows from the fact that, being $S$ dominant, $\mathrm{W}(S)$ is positive. □

Given an out-of-sample vertex $i$ and a class $S$ such that rule (6) holds, we now provide an approximation of the degree of participation of $i$ in $S \cup \{i\}$ which, as pointed out in the previous section, is given by the ratio between $\mathrm{w}_{S \cup \{i\}}(i)$ and $\mathrm{W}(S \cup \{i\})$. This can be used, for example, to get a hard partition of the input data when an instance happens to be assigned to more than one class. By equation (8), we have:

$$\frac{\mathrm{w}_{S \cup \{i\}}(i)}{\mathrm{W}(S \cup \{i\})} = \sum_{h \in S} (\hat{a}_{hi} - a_{hj}) x_h^S \frac{\mathrm{W}(S)}{\mathrm{W}(S \cup \{i\})}$$

for any $j \in S$. Since computing the exact value of the ratio $\mathrm{W}(S)/\mathrm{W}(S \cup \{i\})$ would be computationally expensive, we now provide simple approximation formulas. Since $S$ is dominant, it is reasonable to assume that all weights within it are close to each other. Hence, we approximate $S$ with a clique having constant weight $a$, and impose that it has the same cohesiveness value $f(\mathbf{x}^S) = \mathbf{x}^S \cdot A\mathbf{x}^S$ as the original dominant set. After some algebra, we get

$$a = \frac{|S|}{|S| - 1} f(\mathbf{x}^S)$$

which yields $\mathrm{W}(S) \approx |S| a^{|S|-1}$. Approximating $\mathrm{W}(S \cup \{i\})$ with $|S + 1| a^{|S|}$ in a similar way, we get:

$$\frac{\mathrm{W}(S)}{\mathrm{W}(S \cup \{i\})} \approx \frac{|S| a^{|S|-1}}{|S + 1| a^{|S|}} = \frac{1}{f(\mathbf{x}^S)} \frac{|S| - 1}{|S| + 1}$$

which finally yields:

$$\frac{\mathrm{w}_{S \cup \{i\}}(i)}{\mathrm{W}(S \cup \{i\})} \approx \frac{|S| - 1}{|S| + 1} \left( \frac{\sum_{h \in S} \hat{a}_{hi} x_h^S}{f(\mathbf{x}^S)} - 1 \right). \tag{9}$$

Using the above formula one can easily get, by normalization, an approximation of the characteristic vector $\mathbf{x}^{\hat{S}} \in \Delta_{n+k}$ of $\hat{S}$, the extension of cluster $S$ obtained applying rule (6):

$$\hat{S} = S \cup \{i \in \hat{V} \setminus V : \mathrm{w}_{S \cup \{i\}}(i) > 0\}.$$

With an approximation of $\mathbf{x}^{\hat{S}}$ at hand, it is also easy to compute an approximation of the cohesiveness of the new cluster $\hat{S}$, i.e., $\mathbf{x}^{\hat{S}} \cdot \hat{A}\mathbf{x}^{\hat{S}}$. Indeed, assuming that $\hat{S}$ is dominant in $\hat{G}$, and recalling the KKT equality conditions for program (4) [7], we get $(\hat{A}\mathbf{x}^{\hat{S}})_i = \mathbf{x}^{\hat{S}} \cdot \hat{A}\mathbf{x}^{\hat{S}}$ for all $i \in \hat{S}$. It is therefore natural to approximate the cohesiveness of $\hat{S}$ as a weighted average of the $(\hat{A}\mathbf{x}^{\hat{S}})_i$'s.

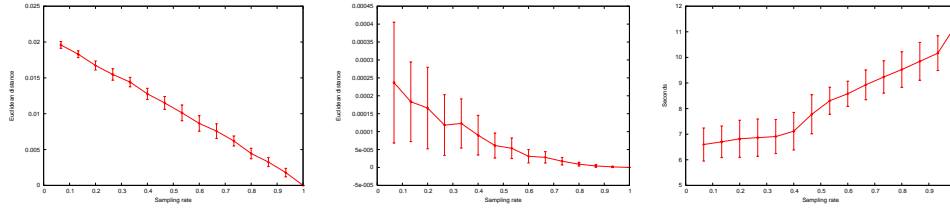

Figure 2: Evaluating the quality of our approximations on a 150-point cluster. Average distance between approximated and actual cluster membership (left) and cohesiveness (middle) as a function of sampling rate. Right: average CPU time as a function of sampling rate.

## 4  Experimental Results

In an attempt to evaluate how the approximations given at the end of the previous section actually compare to the solutions obtained on the dense problem, we conducted the following preliminary experiment. We generated 150 points on the plane so as to form a dominant set (we used a standard Gaussian kernel to obtain similarities), and extracted random samples with increasing sampling rate, ranging from 1/15 to 1. For each sampling rate 100 trials were made, for each of which we computed the Euclidean distance between the approximated and the actual characteristic vector (i.e., cluster membership), as well as the distance between the approximated and the actual cluster cohesiveness (that is, the value of the objective function $f$). Fig. 2 shows the average results obtained. As can be seen, our approximations work remarkably well: with a sampling rate less than 10 % the distance between the characteristic vectors is around 0.02 and this distance decreases linearly towards zero. As for the objective function, the results are even more impressive as the distance from the exact value (i.e., 0.989) rapidly goes to zero starting from 0.00025, at less than 10% rate. Also, note how the CPU time increases linearly as the sampling rate approaches 100%.

Next, we tested our algorithm over the Johns Hopkins University ionosphere database[2] which contains 351 labeled instances from two different classes. As in the previous experiment, similarities were computed using a Gaussian kernel. Our goal was to test how the solutions obtained on the sampled graph compare with those of the original, dense problem and to study how the performance of the algorithm scales w.r.t. the sampling rate. As before, we used sampling rates from 1/15 to 1, and for each such value 100 random samples were extracted. After the grouping process, the out-of-sample instances were assigned to one of the two classes found using rule (6). Then, for each example in the dataset a "success" was recorded whenever the actual class label of the instance coincided with the majority label of its assigned class. Fig. 3 shows the average results obtained. At around 40% rate the algorithm was already able to obtain a classification accuracy of about 73.4%, which is even slightly higher that the one obtained on the dense (100% rate) problem, which is 72.7%. Note that, as in the previous experiment, the algorithm appears to be robust with respect to the choice of the sample data. For the sake of comparison we also ran normalized cut on the *whole* dataset, and it yielded a classification rate of 72.4%.

Finally, we applied our algorithm to the segmentation of brightness images. The image to be segmented is represented as a graph where vertices correspond to pixels and edge-weights reflect the "similarity" between vertex pairs. As customary, we defined a similarity measure between pixels based on brightness proximity. Specifically, following [7], similarity between pixels $i$ and $j$ was measured by $w(i,j) = \exp\left((I(i) - I(j))^2/\sigma^2\right)$ where $\sigma$ is a positive real number which affects the decreasing rate of $w$, and $I(i)$ is defined as the (normalized) intensity value at node $i$. After drawing a set of pixels at random with sampling rate $p = 0.005$, we iteratively found a dominant set in the sampled graph using replicator dynamics [7, 14], we removed it from the graph. and we then employed rule (6)

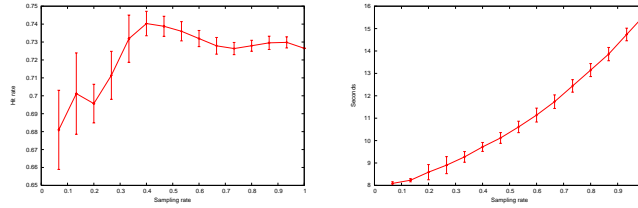

Figure 3: Results on the ionosphere database. Average classification rate (left) and CPU time (right) as a function of sampling rate.

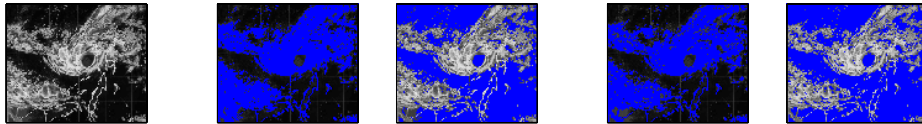

Figure 4: Segmentation results on a $115 \times 97$ weather radar image. From left to right: original image, the two regions found on the sampled image (sampling rate = 0.5%), and the two regions obtained on the whole image (sampling rate = 100%).

to extend it with out-of-sample pixels.

Figure 4 shows the results obtained on a $115 \times 97$ weather radar image, used in [13, 7] as an instance whereby edge-detection-based segmentation would perform poorly. Here, and in the following experiment, the major components of the segmentations are drawn on a blue background. The leftmost cluster is the one obtained after the first iteration of the algorithm, and successive clusters are shown left to right. Note how the segmentation obtained over the sparse image, sampled at 0.5% rate, is almost identical to that obtained over the whole image. In both cases, the algorithms correctly discovered a background and a foreground region. The approximation algorithm took a couple of seconds to return the segmentation, i.e., 15 times faster than the one run over the entire image. Note that our results are better than those obtained with normalized cut, as the latter provides an over-segmented solution (see [13]).

Fig. 5 shows results on two $481 \times 321$ images taken from the Berkeley database.[3] On these images the sampling process produced a sample with no more than $1000$ pixels, and our current MATLAB implementation took only a few seconds to return a solution. Running the grouping algorithm on the whole images (which contain more than $150,000$ pixels) would simply be unfeasible. In both cases, our approximation algorithm partitioned the images into meaningful and clean components. We also ran normalized cut on these images (using the same sample rate of $0.5\%$) and the results, obtained after a long tuning process, confirm its well-known inherent tendency to over-segment the data (see Fig. 5).

## 5   Conclusions

We have provided a simple and efficient extension to the dominant-set clustering framework to deal with the grouping of out-of-sample data. This makes the approach applicable to very large grouping problems, such as high-resolution image segmentation, where it would otherwise be impractical. Experiments show that the solutions extrapolated from the sparse data are comparable with those of the dense problem, which in turn compare favorably with spectral solutions such as normalized cut's, and are obtained in much less time.

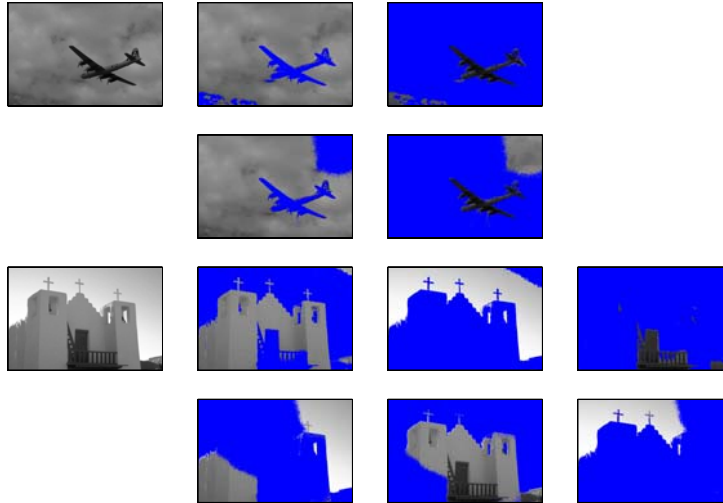

Figure 5: Segmentation results on two $481 \times 321$ images. Left columns: original images. For each image, the first line shows the major regions obtained with our approximation algorithm, while the second line shows the results obtained with normalized cut.

## Footnotes

[1] Observe that $w_S(i)$ depends only on the the weights on the edges of the subgraph induced by $S$. Hence, no ambiguity arises as to whether $w_S(i)$ is computed on $G$ or on $\hat{G}$.

[2]http://www.ics.uci.edu/~mlearn/MLSummary.html

[3]http://www.cs.berkeley.edu/projects/vision/grouping/segbench

# References

[1] Y. Bengio, J.-F. Paiement, P. Vincent, O. Delalleau, N. Le Roux, and M. Ouimet. Out-of-sample extensions for LLE, Isomap, MDS, eigenmaps, and spectral clustering. In: S. Thrun, L. Saul, and B.Schölkopf (Eds.), *Advances in Neural Information Processing Systems 16*, MIT Press, Cambridge, MA, 2004.

[2] C. Fowlkes, S. Belongie, F. Chun, and J. Malik. Spectral grouping using the Nyström method. *IEEE Trans. Pattern Anal. Machine Intell.* 26:214–225, 2004.

[3] T. Hofmann and J. M. Buhmann. Pairwise data clustering by deterministic annealing. *IEEE Trans. Pattern Anal. Machine Intell.* 19:1–14, 1997.

[4] D. Luenberger. *Linear and Nonlinear Programming*. Addison-Wesley, Reading, MA, 1984.

[5] T. S. Motzkin and E. G. Straus. Maxima for graphs and a new proof of a theorem of Turán. *Canad. J. Math.* 17:533–540, 1965.

[6] A. Y. Ng, M. I. Jordan, and Y. Weiss. On spectral clustering: Analysis and an algorithm. In: T. G. Dietterich, S. Becker, and Z. Ghahramani (Eds.), *Advances in Neural Information Processing Systems 14*, MIT Press, Cambridge, MA, pp. 849–856, 2002.

[7] M. Pavan and M. Pelillo. A new graph-theoretic approach to clustering and segmentation. In *Proc. IEEE Conf. Computer Vision and Pattern Recognition*, pp. 145–152, 2003.

[8] M. Pavan, M. Pelillo. Unsupervised texture segmentation by dominant sets and game dynamics. In *Proc. 12th Int. Conf. on Image Analysis and Processing*, pp. 302–307, 2003.

[9] M. Pavan and M. Pelillo. Dominant sets and hierarchical clustering. In *Proc. 9th Int. Conf. on Computer Vision*, pp. 362–369, 2003.

[10] P. Perona and W. Freeman. A factorization approach to grouping. In: H. Burkhardt and B. Neumann (Eds.), *Computer Vision—ECCV'98*, pp. 655–670. Springer, Berlin, 1998.

[11] V. Roth, J. Laub, M. Kawanabe, and J. M. Buhmann. Optimal cluster preserving embedding of nonmetric proximity data. *IEEE Trans. Pattern Anal. Machine Intell.* 25:1540–1551, 2003.

[12] S. Sarkar and K. Boyer. Quantitative measures of change based on feature organization: Eigenvalues and eigenvectors. *Computer Vision and Image Understanding* 71:110–136, 1998.

[13] J. Shi and J. Malik. Normalized cuts and image segmentation. *IEEE Trans. Pattern Anal. Machine Intell.* 22:888–905, 2000.

[14] J. W. Weibull. *Evolutionary Game Theory*. MIT Press, Cambridge, MA, 1995.

[15] Y. Weiss. Segmentation using eigenvectors: A unifying view. In *Proc. 7th Int. Conf. on Computer Vision*, pp. 975–982, 1999.
